# NEURAL NETWORKS FOR MODEL MATCHING AND PERCEPTUAL ORGANIZATION

| Gene Gindi | Eric Mjolsness | P. Anandan |
|---|---|---|
| EE Department | CS Department | CS Department |
| Yale University | Yale University | Yale University |
| New Haven, CT 06520 | New Haven, CT 06520 | New Haven, CT 06520 |

## ABSTRACT

We introduce an optimization approach for solving problems in computer vision that involve multiple levels of abstraction. Our objective functions include compositional and specialization hierarchies. We cast vision problems as inexact graph matching problems, formulate graph matching in terms of constrained optimization, and use analog neural networks to perform the optimization. The method is applicable to perceptual grouping and model matching. Preliminary experimental results are shown.

## 1 Introduction

The minimization of objective functions is an attractive way to formulate and solve visual recognition problems. Such formulations are parsimonious, being expressible in several lines of algebra, and may be converted into artificial neural networks which perform the optimization. Advantages of such networks including speed, parallelism, cheap analog computing, and biological plausibility have been noted [Hopfield and Tank, 1985].

According to a common view of computational vision, recognition involves the construction of abstract descriptions of data governed by a data base of *models*. Abstractions serve as reduced descriptions of complex data useful for reasoning about the objects and events in the scene. The models indicate what objects and properties may be expected in the scene. The complexity of visual recognition demands that the models be organized into compositional hierarchies which express object-part relationships and specialization hierarchies which express object-class relationships. In this paper, we describe a methodology for expressing model-based visual recognition as the constrained minimization of an objective function. Model-specific objective functions are used to govern the dynamic grouping of image elements into recognizable wholes. Neural networks are used to carry out the minimization.

[0] This work was supported in part by AFOSR grant F49620-88-C-0025, and by DARPA grant DAAA15-87-K-0001, by ONR grant N00014-86-0310.

Previous work on optimization in vision has typically been restricted to computations occuring at a single of level of abstraction and/or involving a single model [Barrow and Popplestone, 1971, Hummel and Zucker, 1983, Terzopoulos, 1986]. For example, surface interpolation schemes, even when they include discontinuities [Terzopoulos, 1986] do not include explicit models for physical objects whose surface characteristics determine the expected degree of smoothness. By contrast, heterogeneous and hierarchical model-bases often occur in non-optimization approaches to visual recognition [Hanson and Riseman, 1986] including some which use neural networks [Ballard, 1986]. We attempt to obtain greater expressability and efficiency by incorporating hierarchies of abstraction into the optimization paradigm.

# 2   Casting Model Matching as Optimization

We consider a type of objective function which, when minimized by a neural network, is capable of expressing many of the ideas found in Frame systems in Artificial Intelligence [Minsky, 1975]. These "Frameville" objective functions [Mjolsness et al., 1988, Mjolsness et al., 1989] are particularly well suited to applications in model-based vision, with frames acting as few-parameter abstractions of visual objects or perceptual groupings thereof. Each frame contains real-valued parameters, pointers to other frames, and pointers to predefined models (e.g. models of objects in the world) which determine what portion of the objective function acts upon a given frame.

## 2.1   Model Matching as Graph Matching

Model matching involves finding a match between a set of frames, ultimately derived from visual data, and the predefined static models. A set of pointers represent object-part relationships between frames, and are encoded as a graph or sparse matrix called *ina*. That is, $ina_{ij} = 0$ unless frame $j$ is "in" frame $i$ as one of its parts, in which case $ina_{ij} = 1$ is a "pointer" from $j$ to $i$. The expected object-part relationships between the corresponding models is encoded as a fixed graph or sparse matrix *INA*. A form of inexact graph-matching is required: *ina* should follow *INA* as much as is consistent with the data.

A sparse match matrix $M$ ($0 \leq M_{\alpha i} \leq 1$) of dynamic variables represents the correspondence between model $\alpha$ and frame $i$. To find the best match between the two graphs one can minimize a simple objective function for this match matrix, due to Hopfield [Hopfield, 1984] (see also [Feldman et al., 1988, Malsburg, 1986]), which just counts the number of consistent rectangles (see Figure 1a):

$$E(M) = -\sum_{\alpha\beta} \sum_{ij} INA_{\alpha\beta} ina_{ij} M_{\alpha i} M_{\beta j}. \qquad (1)$$

This expression may be understood as follows: For model $\alpha$ and frame $i$, the match value $M_{\alpha i}$ is to be increased if the neighbors of $\alpha$ (in the *INA* graph) match to the neighbors of $i$ (in the *ina* graph).

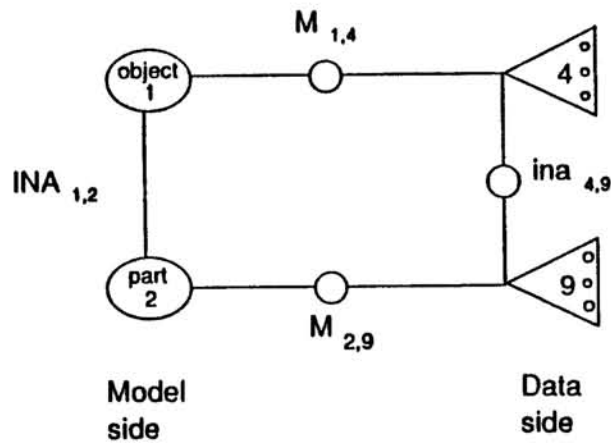

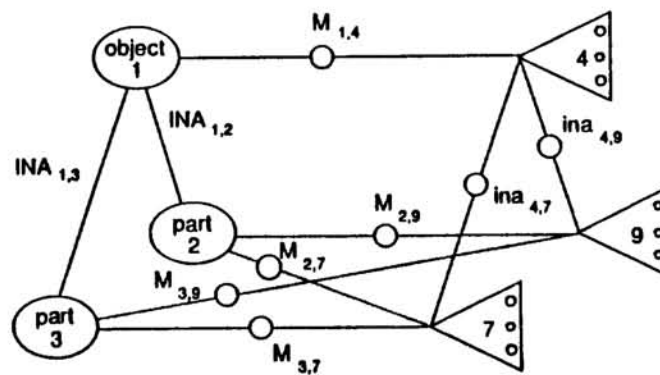

Figure 1: (a) **Examples of Frameville rectangle rule.** Shows the rectangle relationship between frames (triangles) representing a *wing* of a plane, and the *plane* itself. Circles denote dynamic variables, ovals denote models, and triangles denote frames. For the *plane* and *wing* models, the first few parameters of a frame are interpreted as position, length, and orientation. (b) **Frameville sibling competition among parts.** The match variables along the shaded lines ($M_{3,9}$ and $M_{2,7}$) are suppressed in favor of those along the solid lines ($M_{2,9}$ and $M_{3,7}$).

Note that $E(M)$ as defined above can be trivially minimized by setting all the elements of the match matrix to unity. However, to do so will violate additional syntactic constraints of the form $h(M) = 0$ which are imposed on the optimization, either exactly [Platt and Barr, 1988] or as penalty terms [Hopfield and Tank, 1985] $\frac{c}{2}h^2(M)$ added to the objective function. Originally the syntactic constraints simply meant that each frame should match one model and vice versa, as in [Hopfield and Tank, 1985]. But in Frameville, a frame can match both a model and one of its specializations (described later), and a single model can match any number of instances or frames. In addition one can usually formulate constraints stating that if a model matches a frame then two distinct parts of the same model must match two distinct part frames and vice-versa. We have found the following

formulation to be useful:

$$\sum_{\alpha} INA_{\alpha\beta} M_{\alpha i} - \sum_{j} ina_{ij} M_{\beta j} = 0, \quad \forall \beta, i \tag{2}$$

$$\sum_{i} ina_{ij} M_{\alpha i} - \sum_{\beta} INA_{\alpha\beta} M_{\beta j} = 0, \quad \forall \alpha, j \tag{3}$$

where the first sum in each equation is necessary when several high-level models (or frames) share a part. (It turns out that the first sums can be forced to zero or one by other constraints.) The resulting competition is illustrated in Figure 1b. Another constraint is that $M$ should be binary-valued, i.e.,

$$M_{\alpha i}(1 - M_{\alpha i}) = 0, \tag{4}$$

but this constraint can also be handled by a special "analog gain" term in the objective function [Hopfield and Tank, 1985] together with a penalty term $c\sum_{\alpha i} M_{\alpha i}(1 - M_{\alpha i})$.
In Frameville, the *ina* graph actually becomes variable, and is determined by a dynamic grouping or "perceptual organization" process. These new variables require new constraints, starting with $ina_{ij}(1 - ina_{ij}) = 0$, and including many high-level constraints which we now formulate.

## 2.2   Frames and Objective Functions

Frames can be considered as bundles $\vec{F}_i$ of real-valued parameters $F_{ip}$, where $p$ indexes the different parameters of a frame. For efficiency in computing complex arithmetic relationships, such as those involved in coordinate transformations, an analog representation of these parameters is used. A frame contains no information concerning its match criteria or control flow; instead, the match criteria are expressed as objective functions and the control flow is determined by the particular choice of a minimization technique.
In Figure 1a, in order for the rectangle $(1, 4, 9, 2)$ to be consistent, the parameters $F_{4p}$ and $F_{9p}$ should satisfy a criterion dictated by models 1 and 2, such as a restriction on the difference in angles appropriate for a mildly swept back wing. Such a constraint results in the addition of the following term to the objective function:

$$\sum_{i,j,\alpha,\beta} INA_{\alpha\beta} ina_{ij} M_{\alpha i} M_{\beta j} H^{\alpha\beta}(\vec{F}_i, \vec{F}_j) \tag{5}$$

where $H^{\alpha\beta}(\vec{F}_i, \vec{F}_j)$ measures the deviation of the parameters of the data frames from that demanded by the models. The term $H$ can express coordinate transformation arithmetic (e.g. $H^{\alpha\beta}(x_i, x_j) = 1/2[x_i - x_j - \Delta x^{\alpha\beta}]^2$), and its action on a frame $\vec{F}_i$ is selectively controlled or "gated" by $M$ and *ina* variables. This is a fundamental extension of the distance metric paradigm in pattern recognition; because of the complexity of the visual world, we use an entire database of distance metrics $H^{\alpha\beta}$.

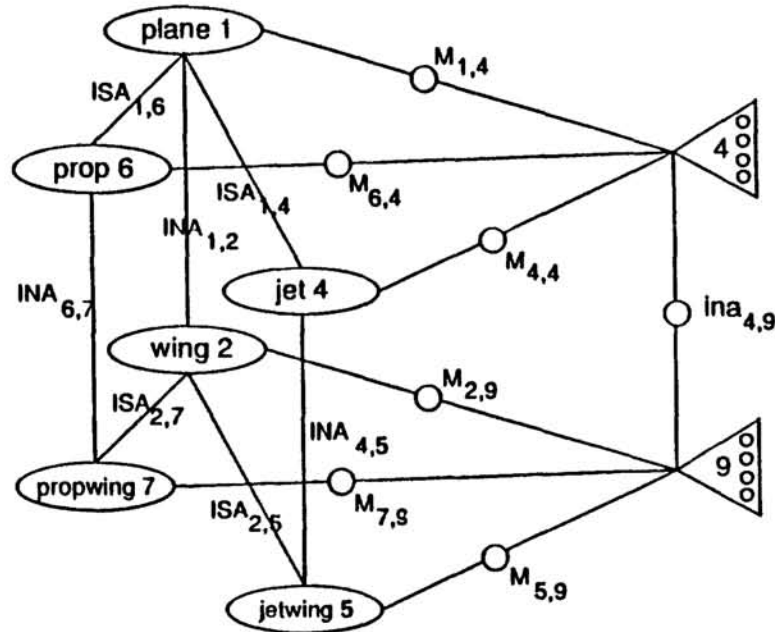

Figure 2: **Frameville specialization hierarchy**. The *plane* model specializes along *ISA* links to a *propeller plane* or a *jet plane* and correspondingly the *wing* model specializes to *prop-wing* or *jet-wing*. Sibling match variables $M_{6,4}$ and $M_{4,4}$ compete as do $M_{7,9}$ and $M_{5,9}$. The winner in these competitions is determined by the consistency of the appropriate rectangles, e.g. if the 4-4-9-5 rectangle is more consistent than the 6-4-9-7 rectangle, then the *jet* model is favored over the *prop* model.

We index the models (and, indirectly, the data base of $H$ metrics) by introducing a static graph of pointers $ISA_{\alpha\beta}$ to act as both a specialization hierarchy and a discrimination network for visual recognition. A frame may simultaneously match to a model and just one of its specializations:

$$M_{\alpha i} - \sum_{\beta} ISA_{\alpha\beta} M_{\beta i} = 0. \tag{6}$$

As a result, *ISA* siblings compete for matches to a given frame (see Figure 2); this competition allows the network to act as a discrimination tree.

Frameville networks have great expressive power, but have a potentially serious problem with cost: for $n$ data frames and $m$ models there may be $O(nm + n^2)$ neurons widely interconnected but sparsely activated. The number of connections is at most the number of monomials in the polynomial objective function, namely $n^2 mf$, where $f$ is the fan-out of the *INA* graph. One solution to the cost problem, used in the line grouping experiments reported in [Mjolsness et al., 1989], is to restrict the flexibility of the frame system by setting most $M$ and *ina* neurons to zero permanently. The few remaining variables can form an efficient data structure

such as a pyramid in vision. A more flexible solution might enforce the sparseness constraints on the $M$ and *ina* neurons during minimization, as well as at the fixed point. Then large savings could result from using "virtual" neurons (and connections) which are created and destroyed dynamically. This and other cost-cutting methods are a subject of continuing research.

# 3   Experimental Results

We describe here experiments involving the recognition of simple stick figures. (Other experiments involving the perceptual grouping of lines are reported in [Mjolsness et al., 1989].) The input data (Figure 3(a)) are line segments parameterized by location $x, y$ and orientation $\theta$, corresponding to frame parameters $F_{jp}$ ($p = 1, 2, 3$). As seen in Figure 3(b), there are two high-level models, "T" and "L" junctions, each composed of three low-level segments. The task is to recognize instances of "T", "L", and their parts, in a translation-invariant manner.

The parameter check term $H^{\alpha\beta}$ of Equation 5 achieves translation invariance by checking the location and orientation of a given part relative to a designated main part and is given by:

$$H^{\alpha\beta}(\vec{F}_i, \vec{F}_j) = \sum_p (F_{ip} - F_{jp} - \Delta_p^{\alpha\beta})^2 \tag{7}$$

Here $F_{jp}$ and $F_{ip}$ are the slots of a low-level segment frame and a high-level main part, respectively, and the quantity $\Delta_p^{\alpha\beta}$ is model information that stores coordinate differences. (Rotation invariance can also be formulated if a different parameterization is used.) It should be noted that absence of the main part does not preclude recognition of the high-level model.

We used the unconstrained optimization technique in [Hopfield and Tank, 1985] and achieved improved results by including terms demanding that at most one model match a given frame, and that at most one high-level frame include a given low-level frame as its part [Mjolsness et al., 1989].

Figure 3(c) shows results of attempts to recognize the junctions in Figure 3(a). When initialized to random values, the network becomes trapped in unfavorable local minima of the fifth-order objective function. (But with only a *single* high-level model in the database, the system recognizes a shape amid noise.) If, however, the network is given a "hint" in the form of an initial state with mainparts and high-level matches set correctly, the network converges to the correct state.

There is a great deal of unexploited freedom in the design of the model base and its objective functions; there may be good design disciplines which avoid introducing spurious local minima. For example, it may be possible to use *ISA* and *INA* hierarchies to guide a network to the desired local minimum.

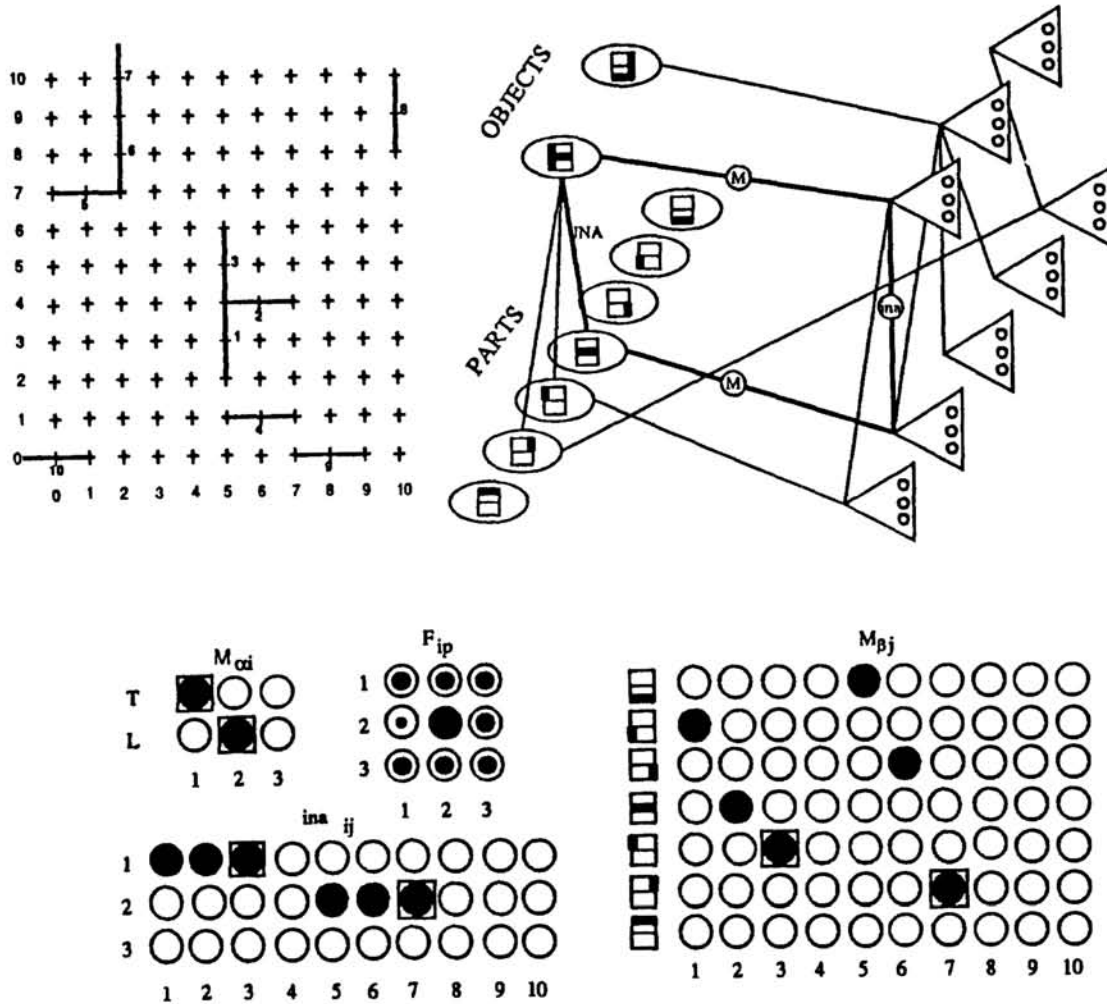

Figure 3: **(a) Input data** consists of unit-length segments oriented horizontally or vertically. The task is translation-invariant recognition of three segments forming a "T" junction (e.g. sticks 1,2,3) or an "L" (e.g. sticks 5,6,7) amid extraneous noise sticks. **(b) Structure of network.** Models occur at two levels. *INA* links are shown for a "T". Each frame has three parameters: position $x, y$ and orientation $\theta$. Also shown are some match and *ina* links. The bold lines highlight a possible consistency rectangle. **(c) Experimental result.** The value of each dynamical variable is displayed as the relative area of the shaded portion of a circle. Matrix $M_{\beta j}$ indicates low-level matches and $M_{\alpha i}$ indicates high-level matches. Grouping of low-level to high-level frames is indicated by the *ina* matrix. The parameters of the high-level frames are displayed in the matrix $F_{ip}$ of linear analog neurons. (The parameters of the low-level frames, held fixed, are not displayed.) The few neurons circumscribed by a square, corresponding to correct matches for the main parts of each model, are clamped to a value near unity. Shaded circles indicate the final correct state.

# 4    Conclusion

Frameville provides opportunities for integrating all levels of vision in a uniform notation which yields analog neural networks. Low-level models such as fixed convolution filters just require analog arithmetic for frame parameters, which is provided. High-level vision typically requires structural matching, also provided. Qualitatively different models may be integrated by specifying their interactions, $H^{\alpha\beta}$.

**Acknowledgements**

We thank J. Utans, J. Ockerbloom and C. Garrett for the Frameville simulations.

**References**

[1] Dana Ballard. Cortical connections and parallel processing: structure and function. *Behavioral and Brain Sciences*, vol 9:67–120, 1986.

[2] Harry G. Barrow and R. J. Popplestone. Relational descriptions in picture processing. In D. Mitchie, editor, *Machine Intelligence 6*, Edinborough University Press, 1971.

[3] Jerome A. Feldman, Mark A. Fanty, and Nigel H. Goddard. Computing with structured neural networks. *IEEE Computer*, 91, March 1988.

[4] Allen R. Hanson and E. M. Riseman. A methodology for the development of general knowledge-based vision systems. In M. A. Arbib and A. R. Hanson, editors, *Vision, Brain, and Cooperative Computation*, MIT Press, 1986.

[5] J. J. Hopfield. Personal communication. October 1984.

[6] J. J. Hopfield and D. W. Tank. 'Neural' computation of decisions in optimization problems. *Biological Cybernetics*, vol. 52:141–152, 1985.

[7] Robert A. Hummel and S. W. Zucker. On the foundations of relaxation labeling processes. *IEEE Transactions on PAMI*, vol. PAMI-5:267–287, May 1983.

[8] Marvin L. Minsky. A framework for representing knowledge. In P. H. Winston, editor, *The Psychology of Computer Vision*, McGraw-Hill, 1975.

[9] Eric Mjolsness, Gene Gindi, and P. Anandan. *Optimization in Model Matching and Perceptual Organization: A First Look*. Technical Report YALEU/DCS/RR-634, Yale University, June 1988.

[10] Eric Mjolsness, Gene Gindi, and P. Anandan. Optimization in Model Matching and Perceptual Organization. *Neural Computation*, to appear.

[11] John C. Platt and Alan H. Barr. Constraint methods for flexible models. *Computer Graphics*, 22(4), August 1988. Proceedings of SIGGRAPH '88.

[12] Demitri Terzopoulos. Regularization of inverse problems involving discontinuities. *IEEE Transactions on PAMI*, vol. PAMI-8:413–424, 1986.

[13] Christoph von der Malsburg and Elie Bienenstock. Statistical coding and short-term synaptic plasticity: a scheme for knowledge representation in the brain. In *Disordered Systems and Biological Organization*, pages 247–252, Springer-Verlag, 1986.